# Discriminative Direction for Kernel Classifiers

**Polina Golland**
Artificial Intelligence Lab
Massachusetts Institute of Technology
Cambridge, MA 02139
*polina@ai.mit.edu*

## Abstract

In many scientific and engineering applications, detecting and under-
standing differences between two groups of examples can be reduced
to a classical problem of training a classifier for labeling new examples
while making as few mistakes as possible. In the traditional classifi-
cation setting, the resulting classifier is rarely analyzed in terms of the
properties of the input data captured by the discriminative model. How-
ever, such analysis is crucial if we want to understand and visualize the
detected differences. We propose an approach to interpretation of the sta-
tistical model in the original feature space that allows us to argue about
the model in terms of the relevant changes to the input vectors. For each
point in the input space, we define a discriminative direction to be the
direction that moves the point towards the other class while introducing
as little irrelevant change as possible with respect to the classifier func-
tion. We derive the discriminative direction for kernel-based classifiers,
demonstrate the technique on several examples and briefly discuss its use
in the statistical shape analysis, an application that originally motivated
this work.

## 1 Introduction

Once a classifier is estimated from the training data, it can be used to label new examples,
and in many application domains, such as character recognition, text classification and oth-
ers, this constitutes the final goal of the learning stage. The statistical learning algorithms
are also used in scientific studies to detect and analyze differences between the two classes
when the "correct answer" is unknown, and the information we have on the differences
is represented implicitly by the training set. Example applications include morphologi-
cal analysis of anatomical organs (comparing organ shape in patients vs. normal controls),
molecular design (identifying complex molecules that satisfy certain requirements), etc. In
such applications, interpretation of the resulting classifier in terms of the original feature
vectors can provide an insight into the nature of the differences detected by the learning
algorithm and is therefore a crucial step in the analysis. Furthermore, we would argue that
studying the spatial structure of the data captured by the classification function is important
in any application, as it leads to a better understanding of the data and can potentially help
in improving the technique.

This paper addresses the problem of translating a classifier into a different representation

that allows us to visualize and study the differences between the classes. We introduce and derive a so called *discriminative direction* at every point in the original feature space with respect to a given classifier. Informally speaking, the discriminative direction tells us how to change any input example to make it look more like an example from another class without introducing any irrelevant changes that possibly make it more similar to other examples from the same class. It allows us to characterize differences captured by the classifier and to express them as changes in the original input examples.

This paper is organized as follows. We start with a brief background section on kernel-based classification, stating without proof the main facts on kernel-based SVMs necessary for derivation of the discriminative direction. We follow the notation used in [3, 8, 9]. In Section 3, we provide a formal definition of the discriminative direction and explain how it can be estimated from the classification function. We then present some special cases, in which the computation can be simplified significantly due to a particular structure of the kernel. Section 4 demonstrates the discriminative direction for different kernels, followed by an example from the problem of statistical analysis of shape differences that originally motivated this work.

## 2   Basic Notation

Given a training set of $l$ pairs $\{(\mathbf{x}_k, y_k)\}_{k=1}^l$, where $\mathbf{x}_k \in \mathbb{R}^n$ are observations and $y_k \in \{-1, 1\}$ are corresponding labels, and a kernel function $K : \mathbb{R}^n \times \mathbb{R}^n \mapsto \mathbb{R}$, (with its implied mapping function $\Phi_K : \mathbb{R}^n \mapsto \mathbb{F}$), the Support Vector Machines (SVMs) algorithm [8] constructs a classifier by implicitly mapping the training data into a higher dimensional space and estimating a linear classifier in that space that maximizes the margin between the classes (Fig. 1a). The normal to the resulting separating hyperplane is a linear combination of the training data:

$$\mathbf{w} = \sum\nolimits_k \alpha_k y_k \Phi_K(\mathbf{x}_k), \tag{1}$$

where the coefficients $\alpha_k$ are computed by solving a constrained quadratic optimization problem. The resulting classifier

$$f_K(\mathbf{x}) = \langle \mathbf{x} \cdot \mathbf{w} \rangle + b = \sum\nolimits_k \alpha_k y_k \langle \Phi_K(\mathbf{x}) \cdot \Phi_K(\mathbf{x}_k) \rangle + b = \sum\nolimits_k \alpha_k y_k K(\mathbf{x}, \mathbf{x}_k) + b \tag{2}$$

defines a nonlinear separating boundary in the original feature space.

## 3   Discriminative Direction

Equations (1) and (2) imply that the classification function $f_K(\mathbf{x})$ is directly proportional to the signed distance from the input point to the separating boundary computed in the higher dimensional space defined by the mapping $\Phi_K$. In other words, the function output depends only on the projection of vector $\Phi_K(\mathbf{x})$ onto $\mathbf{w}$ and completely ignores the component of $\Phi_K(\mathbf{x})$ that is perpendicular to $\mathbf{w}$. This suggests that in order to create a displacement of $\Phi_K(\mathbf{x})$ that corresponds to the differences between the two classes, one should change the vector's projection onto $\mathbf{w}$ while keeping its perpendicular component the same. In the linear case, we can easily perform this operation, since we have access to the image vectors, $\Phi_K(\mathbf{x}) = \mathbf{x}$. This is similar to visualization techniques typically used in linear generative modeling, where the data variation is captured using PCA, and new samples are generated by changing a single principal component at a time. However, this approach is infeasible in the non-linear case, because we do not have access to the image vectors $\Phi_K(\mathbf{x})$. Furthermore, the resulting image vector might not even have a source in the original feature space, i.e., there might be no vector in the original space $\mathbb{R}^n$ that maps into the resulting vector in the space $\mathbb{F}$. Our solution is to search for the direction around

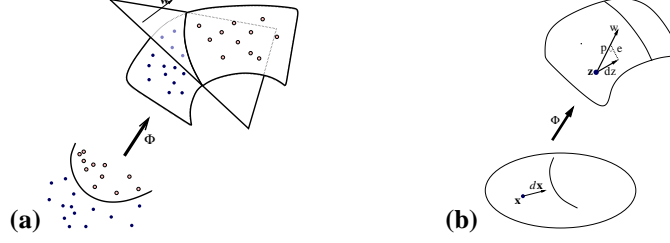

Figure 1: Kernel-based classification (a) and the discriminative direction (b).

the feature vector $\mathbf{x}$ in the original space that minimizes the divergence of its image $\Phi_K(\mathbf{x})$ from the direction of the projection vector $\mathbf{w}$[1]. We call it a *discriminative direction*, as it represents the direction that affects the output of the classifier while introducing as little irrelevant change as possible into the input vector.

Formally, as we move from $\mathbf{x}$ to $\mathbf{x} + d\mathbf{x}$ in $\mathbb{R}^n$, the image vector in the space $\mathbb{F}$ changes by $d\mathbf{z} = \Phi_K(\mathbf{x} + d\mathbf{x}) - \Phi_K(\mathbf{x})$ (Fig. 1b). This displacement can be thought of as a vector sum of its projection onto $\mathbf{w}$ and its deviation from $\mathbf{w}$:

$$\mathbf{p} = \frac{\langle d\mathbf{z} \cdot \mathbf{w}\rangle}{\langle \mathbf{w} \cdot \mathbf{w}\rangle}\mathbf{w} \quad \text{and} \quad \mathbf{e} = d\mathbf{z} - \mathbf{p} = d\mathbf{z} - \frac{\langle d\mathbf{z} \cdot \mathbf{w}\rangle}{\langle \mathbf{w} \cdot \mathbf{w}\rangle}\mathbf{w}. \tag{3}$$

The discriminative direction minimizes the divergence component $\mathbf{e}$, leading to the following optimization problem:

$$\text{minimize} \qquad \mathcal{E}(d\mathbf{x}) = \|\mathbf{e}\|^2 = \langle d\mathbf{z} \cdot d\mathbf{z}\rangle - \frac{\langle d\mathbf{z} \cdot \mathbf{w}\rangle^2}{\langle \mathbf{w} \cdot \mathbf{w}\rangle} \tag{4}$$

$$\text{s.t.} \qquad \|d\mathbf{x}\|^2 = \epsilon. \tag{5}$$

Since the cost function depends only on dot products of vectors in the space $\mathbb{F}$, it can be computed using the kernel function $K$:

$$\langle \mathbf{w} \cdot \mathbf{w}\rangle = \sum_{k,m} \alpha_k \alpha_m y_k y_m K(\mathbf{x}_k, \mathbf{x}_m), \tag{6}$$

$$\langle d\mathbf{z} \cdot \mathbf{w}\rangle = \nabla f_K(\mathbf{x}) d\mathbf{x}, \tag{7}$$

$$\langle d\mathbf{z} \cdot d\mathbf{z}\rangle = d\mathbf{x}^T H_K(\mathbf{x}) d\mathbf{x}, \tag{8}$$

where $\nabla f_K(\mathbf{x})$ is the gradient of the classifier function $f_K$ evaluated at $\mathbf{x}$ and represented by a row-vector and matrix $H_K(\mathbf{x})$ is one of the (equivalent) off-diagonal quarters of the Hessian of $K$, evaluated at $(\mathbf{x}, \mathbf{x})$:

$$H_K(\mathbf{x})[i,j] = \left. \frac{\partial^2 K(\mathbf{u}, \mathbf{v})}{\partial u_i \partial v_j} \right|_{(\mathbf{u}=\mathbf{x}, \mathbf{v}=\mathbf{x})}. \tag{9}$$

Substituting into Equation (4), we obtain

$$\text{minimize} \qquad \mathcal{E}(d\mathbf{x}) = d\mathbf{x}^T \left( H_K(\mathbf{x}) - \|\mathbf{w}\|^{-2} \nabla f_K^T(\mathbf{x}) \nabla f_K(\mathbf{x}) \right) d\mathbf{x} \tag{10}$$

$$\text{s.t.} \qquad \|d\mathbf{x}\|^2 = \epsilon. \tag{11}$$

[1]A similar complication arises in kernel-based generative modeling, e.g., kernel PCA [7]. Constructing linear combinations of vectors in the space $\mathbb{F}$ leads to a global search in the original space [6, 7]. Since we are interested in the direction that best approximates $\mathbf{w}$, we use infinitesimal analysis that results in a different optimization problem.

The solution to this problem is the smallest eigenvector of matrix

$$Q_K(\mathbf{x}) = H_K(\mathbf{x}) - \|\mathbf{w}\|^{-2} \nabla f_K^T(\mathbf{x}) \nabla f_K(\mathbf{x}). \qquad (12)$$

Note that in general, the matrix $Q_K(\mathbf{x})$ and its smallest eigenvector are not the same for different points in the original space and must be estimated separately for every input vector $\mathbf{x}$. Furthermore, each solution defines two opposite directions in the input space, corresponding to the positive and the negative projections onto $\mathbf{w}$. We want to move the input example towards the opposite class and therefore assign the direction of increasing function values to the examples with label $-1$ and the direction of decreasing function values to the examples with label $1$.

Obtaining a closed-form solution of this minimization problem could be desired, or even necessary, if the dimensionality of the input space is high and computing the smallest eigenvector is computationally expensive and numerically challenging. In the next section, we demonstrate how a particular form of the matrix $H_K(\mathbf{x})$ leads to an analytical solution for a large family of kernel functions[2].

### 3.1 Analytical Solution for Discriminative Direction

It is easy to see that if $H_K(\mathbf{x})$ is a multiple of the identity matrix, $H_K(\mathbf{x}) = cI$, then the smallest eigenvector of the matrix $Q_K(\mathbf{x})$ is equal to the largest eigenvector of the matrix $\nabla f_K^T(\mathbf{x}) \nabla f_K(\mathbf{x})$, namely the gradient of the classifier function $\nabla f_K^T(\mathbf{x})$. We will show in this section that both for the linear kernel and, more surprisingly, for RBF kernels, the matrix $H_K(\mathbf{x})$ is of the right form to yield an analytical solution of this form. It is well known that to achieve the fastest change in the value of a function, one should move along its gradient. In the case of the linear and the RBF kernels, the gradient also corresponds to the direction that distinguishes between the two classes while ignoring inter-class variability.

**Dot product kernels,** $K(\mathbf{u}, \mathbf{v}) = k(\langle \mathbf{u} \cdot \mathbf{v} \rangle)$. For any dot product kernel,

$$\left. \frac{\partial^2 K(\mathbf{u}, \mathbf{v})}{\partial u_i \partial v_j} \right|_{(\mathbf{u}=\mathbf{x}, \mathbf{v}=\mathbf{x})} = k'(\|\mathbf{x}\|^2)\delta_{ij} + k''(\|\mathbf{x}\|^2)x_i x_j, \qquad (13)$$

and therefore $H_K(\mathbf{x}) = cI$ for all $\mathbf{x}$ if and only if $k''(\|\mathbf{x}\|^2) \equiv 0$, i.e., when $k$ is a linear function. Thus the linear kernel is the only dot product kernel for which this simplification is relevant. In the linear case, $H_K(\mathbf{x}) = I$, and the discriminative direction is defined as

$$d\mathbf{x}^* = \nabla f_K^T(\mathbf{x}) = \mathbf{w} = \sum \alpha_k y_k \mathbf{x}_k; \quad \mathcal{E}(d\mathbf{x}^*) = 0. \qquad (14)$$

This is not entirely surprising, as the classifier is a linear function in the original space and we can move precisely along $\mathbf{w}$.

Polynomial kernels are a special case of dot product kernels. For polynomials of degree $d \geq 2$,

$$\left. \frac{\partial^2 K(\mathbf{u}, \mathbf{v})}{\partial u_i \partial v_j} \right|_{(\mathbf{u}=\mathbf{x}, \mathbf{v}=\mathbf{x})} = d(1 + \|\mathbf{x}\|^2)^{d-1}\delta_{ij} + d(d-1)(1 + \|\mathbf{x}\|^2)^{d-2}x_i x_j. \qquad (15)$$

$H_K(\mathbf{x})$ is not necessarily diagonal for all $\mathbf{x}$, and we have to solve the general eigenvector problem to identify the discriminative direction.

**Distance kernels, $K(\mathbf{u}, \mathbf{v}) = k(\|\mathbf{u} - \mathbf{v}\|^2)$.** For a distance kernel,

$$\left.\frac{\partial^2 K(\mathbf{u}, \mathbf{v})}{\partial u_i \partial v_j}\right|_{(\mathbf{u}=\mathbf{x}, \mathbf{v}=\mathbf{x})} = -2k'(0)\delta_{ij}, \tag{16}$$

and therefore the discriminative direction can be determined analytically:

$$d\mathbf{x}^* = \nabla f_K^T(\mathbf{x}); \quad \mathcal{E}(d\mathbf{x}^*) = -2k'(0) - \|\mathbf{w}\|^{-2}\|\nabla f_K^T(\mathbf{x})\|^2. \tag{17}$$

The Gaussian kernels are a special case of the distance kernel family, and yield a closed form solution for the discriminative direction:

$$d\mathbf{x}^* = -2/\gamma \sum_k \alpha_k y_k e^{-\frac{\|\mathbf{x}-\mathbf{x}_k\|^2}{\gamma}} (\mathbf{x} - \mathbf{x}_k); \quad \mathcal{E}(d\mathbf{x}^*) = 2/\gamma - \|\nabla f_K^T(\mathbf{x})\|^2/\|\mathbf{w}\|^2. \tag{18}$$

Unlike the linear case, we cannot achieve zero error, and the discriminative direction is only an approximation. The exact solution is unattainable in this case, as it has no corresponding direction in the original space.

## 3.2 Geometric Interpretation

We start by noting that the image vectors $\Phi_K(\mathbf{x})$'s do not populate the entire space $\mathbb{F}$, but rather form a manifold of lower dimensionality whose geometry is fully defined by the kernel function $K$ (Fig. 1). We will refer to this manifold as the *target manifold* in this discussion. We cannot explicitly manipulate elements of the space $\mathbb{F}$, but can only explore the target manifold through search in the original space. We perform the search in the original space by considering all points on an infinitesimally small sphere centered at the original input vector $\mathbf{x}$. In the range space of the mapping function $\Phi_K$, the images of points $\mathbf{x} + d\mathbf{x}$ form an ellipsoid defined by the quadratic form $d\mathbf{z}^T d\mathbf{z} = d\mathbf{x}^T H_K(\mathbf{x}) d\mathbf{x}$. For $H_K(\mathbf{x}) \sim I$, the ellipsoid becomes a sphere, all $d\mathbf{z}$'s are of the same length, and the minimum of error in the displacement vector $d\mathbf{z}$ corresponds to the maximum of the projection of $d\mathbf{z}$ onto $\mathbf{w}$. Therefore, the discriminative direction is parallel to the gradient of the classifier function. If $H_K(\mathbf{x})$ is of any other form, the length of the displacement vector $d\mathbf{z}$ changes as we vary $d\mathbf{x}$, and the minimum of the error in the displacement is not necessarily aligned with the direction that maximizes the projection.

As a side note, our sufficient condition, $H_K(\mathbf{x}) \sim I$, implies that the target manifold is locally flat, i.e., its Riemannian curvature is zero. Curvature and other properties of target manifolds have been studied extensively for different kernel functions [1, 4]. In particular, one can show that the kernel function implies a metric on the original space. Similarly to the natural gradient [2] that maximizes the change in the function value under an arbitrary metric, we minimize the changes that do not affect the function under the metric implied by the kernel.

## 3.3 Selecting Inputs

Given any input example, we can compute the discriminative direction that represents the differences between the two classes captured by the classifier in the neighborhood of the example. But how should we choose the input examples for which to compute the discriminative direction? We argue that in order to study the differences between the classes, one has to examine the input vectors that are close to the separating boundary, namely, the support vectors. Note that this approach is significantly different from the generative modeling, where a "typical" representative, often constructed by computing the mean of the training data, is used for analysis and visualization. In the discriminative framework, we are more interested in the examples that lie close to the opposite class, as they define the differences between the two classes and the optimal separating boundary.

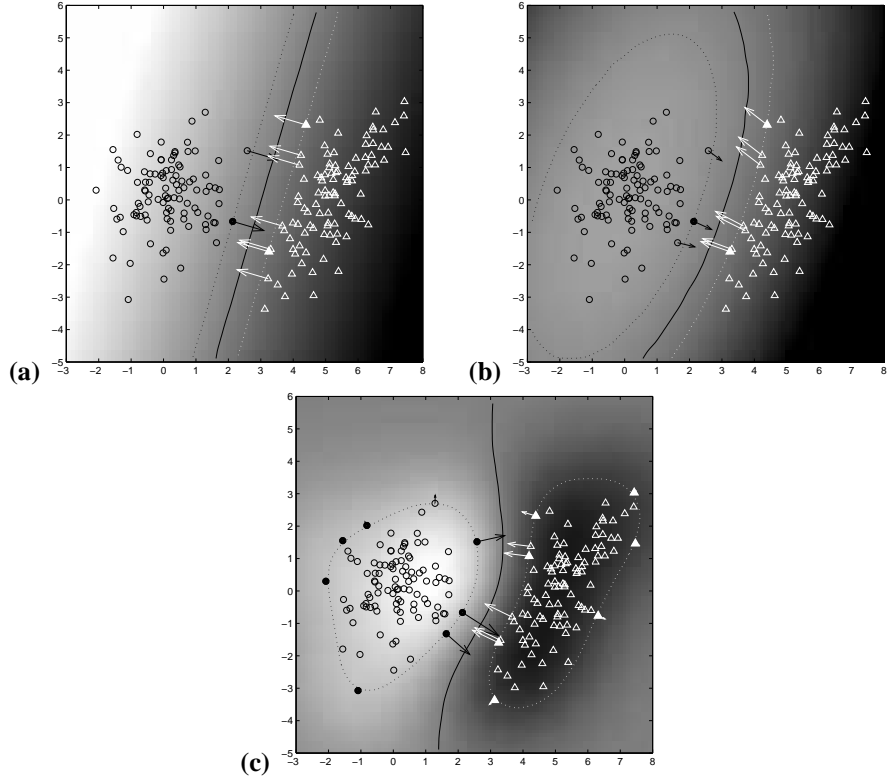

Figure 2: Discriminative direction for linear (a), quadratic (b) and Gaussian RBF (c) classifiers. The background is colored using the values of the classifier function. The black solid line is the separating boundary, the dotted lines indicate the margin corridor. Support vectors are indicated using solid markers. The length of the vectors is proportional to the magnitude of the classifier gradient.

Support vectors define a margin corridor whose shape is determined by the kernel type used for training. We can estimate the distance from any support vector to the separating boundary by examining the gradient of the classification function for that vector. Large gradient indicates that the support vector is close to the separating boundary and therefore can provide more information on the spatial structure of the boundary. This provides a natural heuristic for assigning importance weighting to different support vectors in the analysis of the discriminative direction.

## 4  Simple Example

We first demonstrate the the proposed approach on a simple example. Fig. 2 shows three different classifiers, linear, quadratic and Gaussian RBF, for the same example training set that was generated using two Gaussian densities with different means and covariance matrices. We show the estimated discriminative direction for all points that are close to the separating boundary, not just support vectors. While the magnitude of discriminative direction vector is irrelevant in our infinitesimal analysis, we scaled the vectors in the figure according to the magnitude of the classifier gradient to illustrate importance ranking. Note that for the RBF support vectors far away from the boundary (Fig. 2c), the magnitude of the gradient is so small (tenth of the magnitude at the boundary), it renders the vectors

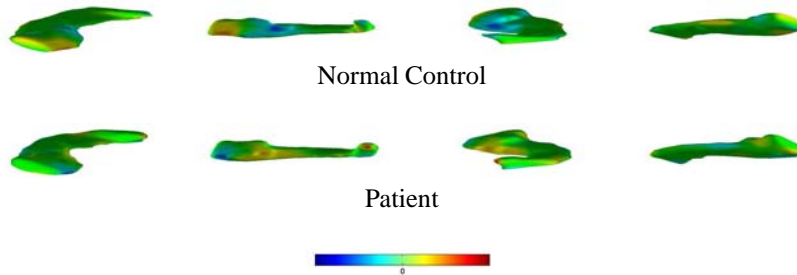

Normal Control

Patient

Figure 3: Right hippocampus in schizophrenia study. First support vector from each group is shown, four views per shape (front, medial, back, lateral). The color coding is used to visualize the amount and the direction of the deformation that corresponds to the discriminative direction, changing from blue (moving inwards) to green (zero deformation) to red (moving outwards).

too short to be visible in the figure. We can see that in the areas where there is enough evidence to estimate the boundary reliably, all three classifiers agree on the boundary and the discriminative direction (lower cluster of arrows). However, if the boundary location is reconstructed based on the regularization defined by the kernel, the classifiers suggest different answers (the upper cluster of arrows), stressing the importance of model selection for classification. The classifiers also provide an indication of the reliability of the differences represented by each arrow, which was repeatedly demonstrated in other experiments we performed.

## 5   Morphological Studies

Morphological studies of anatomical organs motivated the analysis presented in this paper. Here, we show the results for the hippocampus study in schizophrenia. In this study, MRI scans of the brain were acquired for schizophrenia patients and a matched group of normal control subjects. The hippocampus structure was segmented (outlined) in all of the scans. Using the shape information (positions of the outline points), we trained a Gaussian RBF classifier to discriminate between schizophrenia patients and normal controls. However, the classifier in its original form does not provide the medical researchers with information on how the hippocampal shape varies between the two groups. Our goal was to translate the information captured by the classifier into anatomically meaningful terms of organ development and deformation.

In this application, the coordinates in the input space correspond to the surface point locations for any particular example shape. The discriminative direction vector corresponds to displacements of the surface points and can be conveniently represented by a deformation of the original shape, yielding an intuitive description of shape differences for visualization and further analysis. We show the deformation that corresponds to the discriminative direction, omitting the details of shape extraction (see [5] for more information). Fig. 3 displays the first support vector from each group with the discriminative direction "painted" on it. Each row shows four snapshots of the same shape form different viewpoints[3]. The color at every node of the surface encodes the corresponding component of the discriminative direction. Note that the deformation represented by the two vectors is very similar in nature, but of opposite signs, as expected from the analysis in Section 3.3. We can see that the main deformation represented by this pair of vectors is localized in the bulbous "head" of

the structure. The next four support vectors in each group represent a virtually identical deformation to the one shown here. Starting with such visualization, the medical researchers can explore the organ deformation and interaction caused by the disease.

## 6 Conclusions

We presented an approach to quantifying the classifier's behavior with respect to small changes in the input vectors, trying to answer the following question: what changes would make the original input look more like an example from the other class without introducing irrelevant changes? We introduced the notion of the discriminative direction, which corresponds to the maximum changes in the classifier's response while minimizing irrelevant changes in the input. For kernel-based classifiers the discriminative directions is determined by minimizing the divergence of the infinitesimal displacement vector and the normal to the separating hyperplane in the higher dimensional kernel space. The classifier interpretation in terms of the original features in general, and the discriminative direction in particular, is an important component of the data analysis in many applications where the statistical learning techniques are used to discover and study structural differences in the data.

**Acknowledgments.** Quadratic optimization was performed using PR_LOQO optimizer written by Alex Smola. This research was supported in part by NSF IIS 9610249 grant.

## Footnotes

[2]While a very specialized structure of $H_K(\mathbf{x})$ in the next section is sufficient for simplifying the solution significantly, it is by no means necessary, and other kernel families might exist for which estimating the discriminative direction does not require solving the full eigenvector problem.

[3]An alternative way to visualize the same information is to actually generate the animation of the example shape undergoing the detected deformation.

## References

[1] S. Amari and S. Wu. Improving Support Vector Machines by Modifying Kernel Functions. *Neural Networks*, 783-789, 1999.

[2] S. Amari. Natural Gradient Works Efficiently in Learning. *Neural Comp.*, 10:251-276, 1998.

[3] C. J. C. Burges. A Tutorial on Support Vector Machines for Pattern Recognition. *Data Mining and Knowledge Discovery*, 2(2):121-167, 1998.

[4] C. J. C. Burges. Geometry and Invariance in Kernel Based Methods. *In Adv. in Kernel Methods: Support Vector Learning*, Eds. Schölkopf, Burges and Smola, MIT Press, 89-116, 1999.

[5] P. Golland *et al*. Small Sample Size Learning for Shape Analysis of Anatomical Structures. *In Proc. of MICCAI'2000*, LNCS 1935:72-82, 2000.

[6] B. Schölkopf *et al*. Input Space vs. Feature Space in Kernel-Based Methods. *IEEE Trans. on Neural Networks*, 10(5):1000-1017, 1999.

[7] B. Schölkopf, A. Smola, and K.-R. Müller. Nonlinear Component Analysis as a Kernel Eigenvalue Problem. *Neural Comp.*, 10:1299-1319, 1998.

[8] V. N. Vapnik. The Nature of Statistical Learning Theory. *Springer*, 1995.

[9] V. N. Vapnik. Statistical Learning Theory. *John Wiley & Sons*, 1998.
